# Learning with Noise and Regularizers in Multilayer Neural Networks

**David Saad**
Dept. of Comp. Sci. & App. Math.
Aston University
Birmingham B4 7ET, UK
D.Saad@aston.ac.uk

**Sara A. Solla**
AT&T Research Labs
Holmdel, NJ 07733, USA
solla@research.att.com

## Abstract

We study the effect of noise and regularization in an on-line gradient-descent learning scenario for a general two-layer student network with an arbitrary number of hidden units. Training examples are randomly drawn input vectors labeled by a two-layer teacher network with an arbitrary number of hidden units; the examples are corrupted by Gaussian noise affecting either the output or the model itself. We examine the effect of both types of noise and that of weight-decay regularization on the dynamical evolution of the order parameters and the generalization error in various phases of the learning process.

## 1   Introduction

One of the most powerful and commonly used methods for training large layered neural networks is that of on-line learning, whereby the internal network parameters $\{\mathbf{J}\}$ are modified after the presentation of each training example so as to minimize the corresponding error. The goal is to bring the map $f_{\mathbf{J}}$ implemented by the network as close as possible to a desired map $\tilde{f}$ that generates the examples. Here we focus on the learning of continuous maps via gradient descent on a differentiable error function.

Recent work [1]-[4] has provided a powerful tool for the analysis of gradient-descent learning in a very general learning scenario [5]: that of a *student* network with $N$ input units, $K$ hidden units, and a single linear output unit, trained to implement a continuous map from an $N$-dimensional input space $\boldsymbol{\xi}$ onto a scalar $\zeta$. Examples of the target task $\tilde{f}$ are in the form of input-output pairs $(\boldsymbol{\xi}^{\mu}, \zeta^{\mu})$. The output labels $\zeta^{\mu}$ to independently drawn inputs $\boldsymbol{\xi}^{\mu}$ are provided by a *teacher* network of similar

architecture, except that its number $M$ of hidden units is not necessarily equal to $K$.

Here we consider the possibility of a noise process $\rho^{\mu}$ that corrupts the teacher output. Learning from corrupt examples is a realistic and frequently encountered scenario. Previous analysis of this case have been based on various approaches: Bayesian [6], equilibrium statistical physics [7], and nonequilibrium techniques for analyzing learning dynamics [8]. Here we adapt our previously formulated techniques [2] to investigate the effect of different noise mechanisms on the dynamical evolution of the learning process and the resulting generalization ability.

## 2 The model

We focus on a *soft committee machine* [1], for which all hidden-to-output weights are positive and of unit strength. Consider the student network: hidden unit $i$ receives information from input unit $r$ through the weight $J_{ir}$, and its activation under presentation of an input pattern $\boldsymbol{\xi} = (\xi_1, \ldots, \xi_N)$ is $x_i = \mathbf{J}_i \cdot \boldsymbol{\xi}$, with $\mathbf{J}_i = (J_{i1}, \ldots, J_{iN})$ defined as the vector of incoming weights onto the $i$-th hidden unit. The output of the student network is $\sigma(\mathbf{J}, \boldsymbol{\xi}) = \sum_{i=1}^{K} g\left(\mathbf{J}_i \cdot \boldsymbol{\xi}\right)$, where $g$ is the activation function of the hidden units, taken here to be the error function $g(x) \equiv \operatorname{erf}(x/\sqrt{2})$, and $\mathbf{J} \equiv \{\mathbf{J}_i\}_{1 \leq i \leq K}$ is the set of input-to-hidden adaptive weights.

The components of the input vectors $\boldsymbol{\xi}^{\mu}$ are uncorrelated random variables with zero mean and unit variance. Output labels $\zeta^{\mu}$ are provided by a teacher network of similar architecture: hidden unit $n$ in the teacher network receives input information through the weight vector $\mathbf{B}_n = (B_{n1}, \ldots, B_{nN})$, and its activation under presentation of the input pattern $\boldsymbol{\xi}^{\mu}$ is $y_n^{\mu} = \mathbf{B}_n \cdot \boldsymbol{\xi}^{\mu}$. In the noiseless case the teacher output is given by $\zeta_0^{\mu} = \sum_{n=1}^{M} g\left(\mathbf{B}_n \cdot \boldsymbol{\xi}^{\mu}\right)$. Here we concentrate on the architecturally matched case $M = K$, and consider two types of Gaussian noise: additive output noise that results in $\zeta^{\mu} = \rho^{\mu} + \sum_{n=1}^{K} g\left(\mathbf{B}_n \cdot \boldsymbol{\xi}^{\mu}\right)$, and model noise introduced as fluctuations in the activations $y_n^{\mu}$ of the hidden units, $\zeta^{\mu} = \sum_{n=1}^{K} g\left(\rho_n^{\mu} + \mathbf{B}_n \cdot \boldsymbol{\xi}^{\mu}\right)$. The random variables $\rho^{\mu}$ and $\rho_n^{\mu}$ are taken to be Gaussian with zero mean and variance $\sigma^2$.

The error made by a student with weights $\mathbf{J}$ on a given input $\boldsymbol{\xi}$ is given by the quadratic deviation

$$\epsilon(\mathbf{J}, \boldsymbol{\xi}) \equiv \frac{1}{2} \left[ \sigma(\mathbf{J}, \boldsymbol{\xi}) - \zeta_0 \right]^2 = \frac{1}{2} \left[ \sum_{i=1}^{K} g(x_i) - \sum_{n=1}^{K} g(y_n) \right]^2 , \qquad (1)$$

measured with respect to the noiseless teacher (it is also possible to measure performance as deviations with respect to the actual output $\zeta$ provided by the noisy teacher). Performance on a typical input defines the generalization error $\epsilon_g(\mathbf{J}) \equiv \; < \epsilon(\mathbf{J}, \boldsymbol{\xi}) >_{\{\xi\}}$, through an average over all possible input vectors $\boldsymbol{\xi}$ to be performed implicitly through averages over the activations $\mathbf{x} = (x_1, \ldots, x_K)$ and $\mathbf{y} = (y_1, \ldots, y_K)$. These averages can be performed analytically [2] and result in a compact expression for $\epsilon_g$ in terms of *order parameters*: $Q_{ik} \equiv \mathbf{J}_i \cdot \mathbf{J}_k$, $R_{in} \equiv \mathbf{J}_i \cdot \mathbf{B}_n$, and $T_{nm} \equiv \mathbf{B}_n \cdot \mathbf{B}_m$, which represent student-student, student-teacher, and teacher-teacher overlaps, respectively. The parameters $T_{nm}$ are characteristic of the task to be learned and remain fixed during training, while the overlaps $Q_{ik}$ among student hidden units and $R_{in}$ between a student and a teacher hidden units are determined by the student weights $\mathbf{J}$ and evolve during training.

A gradient descent rule on the error made with respect to the actual output provided

by the noisy teacher results in $\mathbf{J}_i^{\mu+1} = \mathbf{J}_i^\mu + \frac{\eta}{N}\, \delta_i^\mu\, \boldsymbol{\xi}^\mu$ for the update of the student weights, where the learning rate $\eta$ has been scaled with the input size $N$, and $\delta_i^\mu$ depends on the type of noise. The time evolution of the overlaps $R_{in}$ and $Q_{ik}$ can be written in terms of similar difference equations. We consider the large $N$ limit, and introduce a normalized number of examples $\alpha = \mu/N$ to be interpreted as a continuous time variable in the $N \to \infty$ limit. The time evolution of $R_{in}$ and $Q_{ik}$ is thus described in terms of first-order differential equations.

## 3   Output noise

The resulting equations of motion for the student-teacher and student-student overlaps are given in this case by:

$$\frac{dR_{in}}{d\alpha} = \eta < \delta_i\, y_n >,\tag{2}$$

$$\frac{dQ_{ik}}{d\alpha} = \eta < \delta_i\, x_k > + \eta < \delta_k\, x_i > + \eta^2 < \delta_i\, \delta_k > + \eta^2 \sigma^2 < g'(x_i)\, g'(x_k) >,$$

where each term is to be averaged over all possible ways in which an example $\boldsymbol{\xi}$ could be chosen at a given time step. These averages have been performed using the techniques developed for the investigation of the noiseless case [2]; the only difference due to the presence of additive output noise is the need to evaluate the fourth term in the equation of motion for $Q_{ik}$, proportional to both $\eta^2$ and $\sigma^2$.

We focus on isotropic uncorrelated teacher vectors: $T_{nm} = T\, \delta_{nm}$, and choose $T = 1$ in our numerical examples. The time evolution of the overlaps $R_{in}$ and $Q_{ik}$ follows from integrating the equations of motion (2) from initial conditions determined by a random initialization of the student vectors $\{\mathbf{J}_i\}_{1 \le i \le K}$. Random initial norms $Q_{ii}$ for the student vectors are taken here from a uniform distribution in the $[0, 0.5]$ interval. Overlaps $Q_{ik}$ between independently chosen student vectors $\mathbf{J}_i$ and $\mathbf{J}_k$, or $R_{in}$ between $\mathbf{J}_i$ and an unknown teacher vector $\mathbf{B}_n$, are small numbers of order $1/\sqrt{N}$ for $N \gg K$, and taken here from a uniform distribution in the $[0, 10^{-12}]$ interval.

We show in Figures 1.a and 1.b the evolution of the overlaps for a noise variance $\sigma^2 = 0.3$ and learning rate $\eta = 0.2$. The example corresponds to $M = K = 3$. The qualitative behavior is similar to the one observed for $M = K$ in the noiseless case extensively analyzed in [2]. A very short transient is followed by a long plateau characterized by lack of differentiation among student vectors: all student vectors have the same norm $Q_{ii} = Q$, the overlap between any two different student vectors takes a unique value $Q_{ik} = C$ for $i \neq k$, and the overlap $R_{in}$ between an arbitrary student vector $i$ and a teacher vector $n$ is independent of $i$ (as student vectors are indistinguishable in this regime) and of $n$ (as the teacher is isotropic), resulting in $R_{in} = R$. This phase is characterized by an unstable symmetric solution; the perturbation introduced through the nonsymmetric initialization of the norms $Q_{ii}$ and overlaps $R_{in}$ eventually takes over in a transition that signals the onset of specialization.

This process is driven by a breaking of the uniform symmetry of the matrix of student-teacher overlaps: each student vector acquires an increasingly dominant overlap $R$ with a specific teacher vector which it begins to imitate, and a gradually decreasing secondary overlap $S$ with the remaining teacher vectors. In the example of Figure 1.b the assignment corresponds to $i = 1 \to n = 1$, $i = 2 \to n = 3$, and $i = 3 \to n = 2$. A relabeling of the student hidden units allows us to identify $R$ with the diagonal elements and $S$ with the off-diagonal elements of the matrix of student-teacher overlaps.

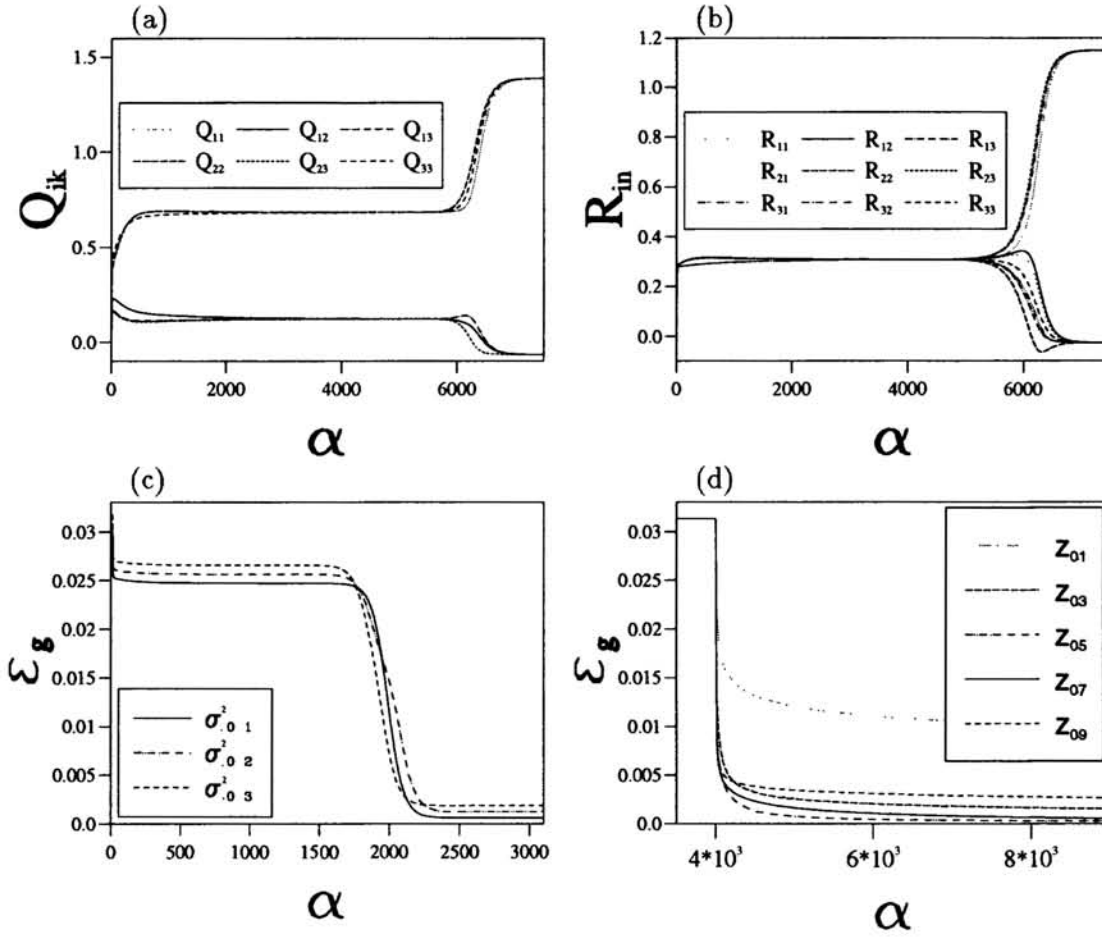

Figure 1: Dependence of the overlaps and the generalization error on the normalized number of examples $\alpha$ for a three-node student learning corrupted examples generated by an isotropic three-node teacher. (a) student-student overlaps $Q_{ik}$ and (b) student-teacher overlaps $R_{in}$ for $\sigma^2 = 0.3$. The generalization error is shown in (c) for different values of the noise variance $\sigma^2$, and in (d) for different powers of the polynomial learning rate decay, focusing on $\alpha > \alpha_0$ (asymptotic regime).

Asymptotically the secondary overlaps $S$ decay to zero, while $R_{in} \to \sqrt{Q_{ii}}$ indicates full alignment for $T_{nn} = 1$. As specialization proceeds, the student weight vectors grow in length and become increasingly uncorrelated. It is interesting to observe that in the presence of noise the student vectors grow asymptotically longer than the teacher vectors: $Q_{ii} \to Q_\infty > 1$, and acquire a small negative correlation with each other. Another detectable difference in the presence of noise is a larger gap between the values of $Q$ and $C$ in the symmetric phase. Larger norms for the student vectors result in larger generalization errors: as shown in Figure 1.c, the generalization error increases monotonically with increasing noise level, both in the symmetric and asymptotic regimes.

For an isotropic teacher, the teacher-student and student-student overlaps can thus be fully characterized by four parameters: $Q_{ik} = Q\delta_{ik} + C(1-\delta_{ik})$ and $R_{in} = R\delta_{in} + S(1-\delta_{in})$. In the symmetric phase the additional constraint $R = S$ reflects the lack of differentiation among student vectors and reduces the number of parameters to three.

The symmetric phase is characterized by a fixed point solution to the equations

of motion (2) whose coordinates can be obtained analytically in the small noise approximation: $R^* = 1/\sqrt{K(2K-1)} + \eta\,\sigma^2\,r_s$ , $Q^* = 1/(2K-1) + \eta\,\sigma^2\,q_s$ , and $C^* = 1/(2K-1) + \eta\,\sigma^2\,c_s$, with $r_s$, $q_s$, and $c_s$ given by relatively simple functions of $K$. The generalization error in this regime is given by:

$$\epsilon_g^* = \frac{K}{\pi}\left(\frac{\pi}{6} - K\arcsin\left(\frac{1}{2K}\right)\right) + \frac{\sigma^2\eta}{2\pi}\frac{(2K-1)^{3/2}}{(2K+1)^{1/2}} \; ; \tag{3}$$

note its increase over the corresponding noiseless value, recovered for $\sigma^2 = 0$.

The asymptotic phase is characterized by a fixed point solution with $R^* \neq S^*$. The coordinates of the asymptotic fixed point can also be obtained analytically in the small noise approximation: $R^* = 1 + \eta\,\sigma^2\,r_a$, $S^* = -\eta\,\sigma^2\,s_a$, $Q^* = 1 + \eta\,\sigma^2\,q_a$, and $C^* = -\eta\,\sigma^2\,c_a$, with $r_a$, $s_a$, $q_a$, and $c_a$ given by rational functions of $K$ with corrections of order $\eta$. The asymptotic generalization error is given by

$$\epsilon_g^* = \frac{\sqrt{3}}{6\pi}\,\eta\,\sigma^2 K \; . \tag{4}$$

Explicit expressions for the coefficients $r_s, q_s, c_s, r_a, s_a, q_a$, and $c_a$ will not be given here for lack of space; suffice it to say that the fixed point coordinates predicted on the basis of the small noise approximation are found to be in excellent agreement with the values obtained from the numerical integration of the equations of motion for $\sigma^2 \leq 0.3$.

It is worth noting in Figure 1.c that in the small noise regime the length of the symmetric plateau decreases with increasing noise. This effect can be investigated analytically by linearizing the equations of motion around the symmetric fixed point and identifying the positive eigenvalue responsible for the escape from the symmetric phase. This calculation has been carried out in the small noise approximation, to obtain $\lambda = (2/\pi)K(2K-1)^{-1/2}(2K+1)^{-3/2} + \lambda_\sigma \sigma^2 \eta$, where $\lambda_\sigma$ is positive and increases monotonically with $K$ for $K > 1$. A faster escape from the symmetric plateau is explained by this increase of the positive eigenvalue. The calculation is valid for $\sigma^2\eta \ll 1$; we observe experimentally that the trend is reversed as $\sigma^2$ increases. A small level of noise assists in the process of differentiation among student vectors, while larger levels of noise tend to keep student vectors equally ignorant about the task to be learned.

The asymptotic value (4) for the generalization error indicates that learning at finite $\eta$ will result in asymptotically suboptimal performance for $\sigma^2 > 0$. A monotonic decrease of the learning rate is necessary to achieve optimal asymptotic performance with $\epsilon_g^* = 0$. Learning at small $\eta$ results in long trapping times in the symmetric phase; we therefore suggest starting the training process with a relatively large value of $\eta$ and switching to a decaying learning rate at $\alpha = \alpha_0$, after specialization begins. We propose $\eta = \eta_0$ for $\alpha \leq \alpha_0$ and $\eta = \eta_0/(\alpha - \alpha_0)^z$ for $\alpha > \alpha_0$. Convergence to the asymptotic solution requires $z \leq 1$. The value $z = 1$ corresponds to the fastest decay for $\eta(\alpha)$; the question of interest is to determine the value of $z$ which results in fastest decay for $\epsilon_g(\alpha)$. Results shown in Figure 1.d for $\alpha > \alpha_0 = 4000$ correspond to $M = K = 3$, $\eta_0 = 0.7$, and $\sigma^2 = 0.1$. Our numerical results indicate optimal decay of $\epsilon_g(\alpha)$ for $z = 1/2$. A rigorous justification of this result remains to be found.

## 4   Model noise

The resulting equations of motion for the student-teacher and student-student overlaps can also be obtained analytically in this case; they exhibit a structure remark-

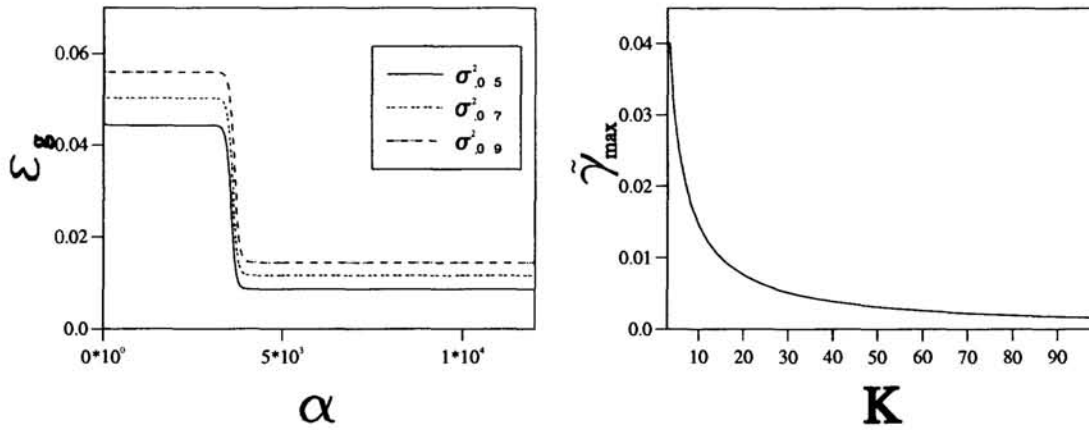

Figure 2: Left - The generalization error for different values of the noise variance $\sigma^2$; training examples are corrupted by model noise. Right - $\tilde{\gamma}_{max}$ as a function of $K$.

ably similar to those for the noiseless case reported in [2], except for some changes in the relevant covariance matrices.

A numerical investigation of the dynamical evolution of the overlaps and generalization error reveals qualitative and quantitative differences with the case of additive output noise: 1) The sensitivity to noise is much higher for model noise than for output noise. 2) The application of independent noise to the individual teacher hidden units results in an effective anisotropic teacher and causes fluctuations in the symmetric phase; the various student hidden units acquire some degree of differentiation and the symmetric phase can no longer be fully characterized by unique values of $Q$ and $C$. 3) The noise level does not affect the length of the symmetric phase.

The effect of model noise on the generalization error is illustrated in Figure 2 for $M = K = 3$, $\eta = 0.2$, and various noise levels. The generalization error increases monotonically with increasing noise level, both in the symmetric and asymptotic regimes, but there is no modification in the length of the symmetric phase. The dynamical evolution of the overlaps, not shown here for the case of model noise, exhibits qualitative features quite similar to those discussed in the case of additive output noise: we observe again a noise-induced widening of the gap between $Q$ and $C$ in the symmetric phase, while the asymptotic phase exhibits an enhancement of the norm of the student vectors and a small degree of negative correlation between them.

Approximate analytic expressions based on a small noise expansion have been obtained for the coordinates of the fixed point solutions which describe the symmetric and asymptotic phases. In the case of model noise the expansions for the symmetric solution are independent of $\eta$ and depend only on $\sigma^2$ and $K$. The coordinates of the asymptotic fixed point can be expressed as: $R^* = 1 + \sigma^2 \, r_a$, $S^* = -\sigma^2 \, s_a$, $Q^* = 1 + \sigma^2 \, q_a$, $C^* = -\sigma^2 \, c_a$, with coefficients $r_a$, $s_a$, $q_a$, and $c_a$ given by rational functions of $K$ with corrections of order $\eta$. The important difference with the output noise case is that the asymptotic fixed point is shifted from its noiseless position even for $\eta = 0$. It is therefore not possible to achieve optimal asymptotic performance even if a decaying learning rate is utilized. The asymptotic generalization error is given by

$$\epsilon_g^* = \frac{\sqrt{3}}{12\pi} \, \sigma^2 K + \eta \, \sigma^2 K \, \epsilon_a(K, \eta) \, . \tag{5}$$

Note that the asymptotic generalization error remains finite even as $\eta \to 0$.

## 5   Regularizers

A method frequently used in real world training scenarios to overcome the effects of noise and parameter redundancy ($K > M$) is the use of regularizers such as weight decay (for a review see [6]).

Weight-decay regularization is easily incorporated within the framework of on-line learning; it leads to a rule for the update of the student weights of the form $\mathbf{J}_i^{\mu+1} = \mathbf{J}_i^{\mu} + \frac{\eta}{N} \delta_i^{\mu} \boldsymbol{\xi}^{\mu} - \frac{\gamma}{N} \mathbf{J}_i^{\mu}$. The corresponding equations of motion for the dynamical evolution of the teacher-student and student-student overlaps can again be obtained analytically and integrated numerically from random initial conditions.

The picture that emerges is basically similar to that described for the noisy case: the dynamical evolution of the learning process goes through the same stages, although specific values for the order parameters and generalization error at the symmetric phase and in the asymptotic regime are changed as a consequence of the modification in the dynamics.

Our numerical investigations have revealed no scenario, either when training from noisy data or in the presence of redundant parameters, where weight decay improves the system performance or speeds up the training process. This lack of effect is probably a generic feature of on-line learning, due to the absence of an additive, stationary error surface defined over a finite and fixed training set. In off-line (batch) learning, regularization leads to improved performance through the modification of such error surface. These observations are consistent with the absence of 'overfitting' phenomena in on-line learning. One of the effects that arises when weight-decay regularization is introduced in on-line learning is a prolongation of the symmetric phase, due to a decrease in the positive eingenvalue that controls the onset of specialization. This positive eigenvalue, which signals the instability of the symmetric fixed point, decreases monotonically with increasing regularization strength $\gamma$, and crosses zero at $\gamma_{max} = \eta \tilde{\gamma}_{max}$. The dependence of $\tilde{\gamma}_{max}$ on $K$ is shown in Figure 2; for $\gamma > \gamma_{max}$ the symmetric fixed point is stable and the system remains trapped there for ever.

The work reported here focuses on an architecturally matched scenario, with $M = K$. Over-realizable cases with $K > M$ show a rich behavior that is rather less amenable to generic analysis. It will be of interest to examine the effects of different types of noise and regularizers in this regime.

**Acknowledgement:** D.S. acknowledges support from EPSRC grant GR/L19232.

## References

[1] M. Biehl and H. Schwarze, *J. Phys. A* **28**, 643 (1995).

[2] D. Saad and S.A. Solla, *Phys. Rev. E* **52**, 4225 (1995).

[3] D. Saad and S.A. Solla, preprint (1996).

[4] P. Riegler and M. Biehl, *J. Phys. A* **28**, L507 (1995).

[5] G. Cybenko, *Math. Control Signals and Systems* **2**, 303 (1989).

[6] C.M. Bishop, *Neural networks for pattern recognition*, (Oxford University Press, Oxford, 1995).

[7] T.L.H. Watkin, A. Rau, and M. Biehl, *Rev. Mod. Phys.* **65**, 499 (1993).

[8] K.R. Müller, M. Finke, N. Murata, K. Schulten, and S. Amari, *Neural Computation* **8**, 1085 (1996).